# Universal Approximation and Learning of Trajectories Using Oscillators

**Pierre Baldi***
Division of Biology
California Institute of Technology
Pasadena, CA 91125
*pfbaldi@juliet.caltech.edu*

**Kurt Hornik**
Technische Universität Wien
Wiedner Hauptstraße 8–10/1071
A-1040 Wien, Austria
*Kurt.Hornik@tuwien.ac.at*

## Abstract

Natural and artificial neural circuits must be capable of traversing specific state space trajectories. A natural approach to this problem is to learn the relevant trajectories from examples. Unfortunately, gradient descent learning of complex trajectories in amorphous networks is unsuccessful. We suggest a possible approach where trajectories are realized by combining simple oscillators, in various modular ways. We contrast two regimes of fast and slow oscillations. In all cases, we show that banks of oscillators with bounded frequencies have universal approximation properties. Open questions are also discussed briefly.

## 1   INTRODUCTION: TRAJECTORY LEARNING

The design of artificial neural systems, in robotics applications and others, often leads to the problem of constructing a recurrent neural network capable of producing a particular trajectory, in the state space of its visible units. Throughout evolution, biological neural systems, such as central pattern generators, have also been faced with similar challenges. A natural approach to tackle this problem is to try to "learn" the desired trajectory, for instance through a process of trial and error and subsequent optimization. Unfortunately, gradient descent learning of complex trajectories in amorphous networks is unsuccessful. Here, we suggest a possible approach where trajectories are realized, in a modular and hierarchical fashion, by combining simple oscillators. In particular, we show that banks of oscillators have universal approximation properties.

To begin with, we can restrict ourselves to the simple case of a network with one[1] visible linear unit and consider the problem of adjusting the network parameters in a way that the output unit activity $u(t)$ is equal to a target function $f(t)$, over an interval of time $[0, T]$. The hidden units of the network may be non-linear and satisfy, for instance, one of the usual neural network charging equations such as

$$\frac{du_i}{dt} = -\frac{u_i}{\tau_i} + \sum_j w_{ij} f_j u_j(t - \tau_{ij}), \tag{1}$$

where $\tau_i$ is the time constant of the unit, the $\tau_{ij}$ represent interaction delays, and the functions $f_j$ are non-linear input/output functions, sigmoidal or other. In the next section, we briefly review three possible approaches for solving this problem, and some of their limitations. In particular, we suggest that complex trajectories can be synthesized by proper combination of simple oscillatory components.

## 2   THREE DIFFERENT APPROACHES TO TRAJECTORY LEARNING

### 2.1   GRADIENT DESCENT APPROACHES

One obvious approach is to use a form of gradient descent for recurrent networks (see [2] for a review), such as back-propagation through time, in order to modify any adjustable parameters of the networks (time constants, delays, synaptic weights and/or gains) to reduce a certain error measure, constructed by comparing the output $u(t)$ with its target $f(t)$. While conceptually simple, gradient descent applied to amorphous networks is not a successful approach, except on the most simple trajectories. Although intuitively clear, the exact reasons for this are not entirely understood, and overlap in part with the problems that can be encountered with gradient descent in simple feed-forward networks on regression or classification tasks.

There is an additional set of difficulties with gradient descent learning of fixed points or trajectories, that is specific to *recurrent* networks, and that has to do with the bifurcations of the system being considered. In the case of a recurrent[2] network, as the parameters are varied, the system may or may not undergo a series of bifurcations, i.e., of abrupt changes in the structure of its trajectories and, in particular, of its attractors (fixed points, limit cycles, . . . ). This in turn may translate into abrupt discontinuities, oscillations or non-convergence in the corresponding learning curve. At each bifurcation, the error function is usually discontinuous, and therefore the gradient is not defined. Learning can be disrupted in two ways: when unwanted abrupt changes occur in the flow of the dynamical system, or when desirable bifurcations are prevented from occurring. A classical example of the second type is the case of a neural network with very small initial weights being trained to oscillate, in a symmetric and stable fashion, around the origin. With small initial weights, the network in general converges to its unique fixed point at the origin, with a large error. If we slightly perturb the weights, remaining away from any bifurcation, the network continues to converge to its unique fixed point which now may be slightly displaced from the origin, and yield an even greater error, so that learning by gradient descent becomes impossible (the starting configuration of zero weights is a local minimum of the error function).

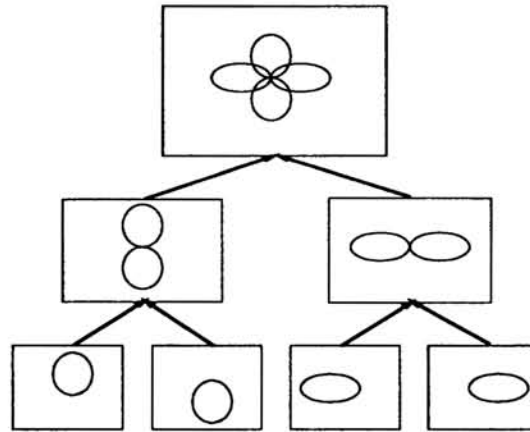

Figure 1: A schematic representation of a 3 layer oscillator network for double figure eight. Oscillators with period $T$ in a given layer gate the corresponding oscillators, with period $T/2$, in the previous layer.

## 2.2 DYNAMICAL SYSTEM APPROACH

In the dynamical system approach, the function $f(t)$ is approximated in time, over $[0, T]$ by a sequence of points $y_0, y_1, \ldots$. These points are associated with the iterates of a dynamical system, i.e., $y_{n+1} = F(y_n) = F^n(y_0)$, for some function $F$. Thus the network implementation requires mainly a feed-forward circuit that computes the function $F$. It has a simple overall recursive structure where, at time $n$, the output $F(y_n)$ is calculated, and fed back into the input for the next iteration. While this approach is entirely general, it leaves open the problem of constructing the function $F$. Of course, $F$ can be learned from examples in a usual feed-forward connectionist network. But, as usual, the complexity and architecture of such a network are difficult to determine in general. Another interesting issue in trajectory learning is how time is represented in the network, and whether some sort of clock is needed. Although occasionally in the literature certain authors have advocated the introduction of an input unit whose output is the time $t$, this explicit representation is clearly not a suitable representation, since the problem of trajectory learning reduces then entirely to a regression problem. The dynamical system approach relies on one basic clock to calculate $F$ and recycle it to the input layer. In the next approach, an implicit representation of time is provided by the periods of the oscillators.

## 2.3 OSCILLATOR APPROACH

A different approach was suggested in [1] where, loosely speaking, complex trajectories are realized using weakly pre-structured networks, consisting of shallow hierarchical combinations of simple oscillatory modules. The oscillatory modules can consist, for instance, of simple oscillator rings of units satisfying Eq. 1, with two or three high-gain neurons, and an odd number of inhibitory connections ([3]).

To fix the ideas, consider the typical test problem of constructing a network capable of producing a trajectory associated with a double figure eight curve (i.e., a set of four loops joined at one point), see Fig. 1. In this example, the first level of the hierarchy could contain four oscillator rings, one for each loop of the target trajectory. The parameters in each one of these four modules can be adjusted, for instance by gradient descent, to match each of the loops in the target trajectory.

The second level of the pyramid should contain two control modules. Each of these modules controls a distinct pair of oscillator networks from the first level, so that each control network in the second level ends up producing a simple figure eight. Again, the control networks in level two can be oscillator rings and their parameters can be adjusted. In particular, after the learning process is completed, they should be operating in their high-gain regimes and have a period equal to the sum of the periods of the circuits each one controls.

Finally, the third layer consists of another oscillatory and adjustable module which controls the two modules in the second level, so as to produce a double figure eight. The third layer module must also end up operating in its high-gain regime with a period equal to four times the period of the oscillators in the first layer. In general, the final output trajectory is also a limit cycle because it is obtained by superposition of limit cycles in the various modules. If the various oscillators relax to their limit cycles independently of one another, it is essential to provide for adjustable delays between the various modules in order to get the proper phase adjustments. In this way, a sparse network with 20 units or so can be constructed that can successfully execute a double figure eight.

There are actually different possible neural network realizations depending on how the action of the control modules is implemented. For instance, if the control units are gating the connections between corresponding layers, this amounts to using higher order units in the network. If one high-gain oscillatory unit, with activity $c(t)$ always close to 0 or 1, gates the oscillatory activities of two units $u_1(t)$ and $u_2(t)$ in the previous layer, then the overall output can be written as

$$\text{out}(t) = c(t)u_1(t) + (1 - c(t))u_2(t). \tag{2}$$

The number of layers in the network then becomes a function of the order of the units one is willing to use. This approach could also be described in terms of a dynamic mixture of experts architecture, in its high gain regime. Alternatively, one could assume the existence of a fast weight dynamics on certain connections governed by a corresponding set of differential equations. Although we believe that oscillators with limit cycles present several attractive properties (stability, short transients, biological relevance, ...), one can conceivably use completely different circuits as building blocks in each module.

## 3   GENERALIZATION AND UNIVERSAL APPROXIMATION

We have just described an approach that combines a modular hierarchical architecture, together with some simple form of learning, enabling the synthesis of a neural circuit suitable for the production of a double figure eight trajectory. It is clear that the same approach can be extended to triple figure eight or, for that matter, to any trajectory curve consisting of an arbitrary number of simple loops with a common period and one common point. In fact it can be extended to any arbitrary trajectory. To see this, we can subdivide the time interval $[0, T]$ into $n$ equal intervals of duration $\epsilon = T/n$. Given a certain level of required precision, we can always find $n$ oscillator networks with period $T$ (or a fraction of $T$) and visible trajectory $u_i(t)$, such that for each $i$, the $i$-th portion of the trajectory $u(t)$ with $i\epsilon \leq t \leq (i + 1)\epsilon$ can be well approximated by a portion of $u_i(t)$, the trajectory of the $i$-th oscillator. The target trajectory can then be approximated as

$$u(t) \approx \sum_{i=1}^{n} c_i(t)u_i(t). \tag{3}$$

As usual, the control coefficient $c_i(t)$ must have also period $T$ and be equal to 1 for $i\epsilon \le t \le (i+1)\epsilon$, and 0 otherwise. The control can be realized with one large high-gain oscillator, or as in the case described above, by a hierarchy of control oscillators arranged, for instance, as a binary tree of depth $m$ if $n = 2^m$, with the corresponding multiple frequencies.

We can now turn to a slightly different oscillator approach, where trajectories are to be approximated with linear combinations of oscillators, with *constant* coefficients. What we would like to show again is that oscillators are universal approximators for trajectories. In a sense, this is already a well-known result of Fourier theory since, for instance, any reasonable function $f$ with period $T$ can be expanded in the form[3]

$$f(t) = \sum_{k=-\infty}^{\infty} \gamma_k e^{2\pi i \lambda_k t}, \qquad \lambda_k = k/T. \tag{4}$$

For sufficiently smooth target functions, without high frequencies in their spectrum, it is well known that the series in Eq. 4 can be truncated. Notice, however, that both Eqs. 3 and 4 require having component oscillators with relatively high frequencies, compared to the final trajectory. This is not implausible in biological motor control, where trajectories have typical time scales of a fraction of a second, and single control neurons operate in the millisecond range. A rather different situation arises if the component oscillators are "slow" with respect to the final product.

The Fourier representation requires in principle oscillations with arbitrarily large frequencies $(0, 1/T, 2/T, \ldots, n/T, \ldots)$. Most likely, relatively small variations in the parameters (for instance gains, delays and/or synaptic weights) of an oscillator circuit can only lead to relatively small but continuous variations of the overall frequency. For instance, in [3] it is shown that the period $T$ of an oscillator ring with $n$ units obeying Eq. 1 must satisfy

$$2 \left( \sum_i \tau_{ii-1} + \ln 2 \sum_i \tau_i \right) \le T \le 2 \left( \sum_i \tau_{ii-1} + \sum_i \tau_i \right).$$

Thus, we need to show that a decomposition similar in flavor to Eq. 4 is possible, but using oscillators with frequencies in a bounded interval. Notice that by varying the parameters of a basic oscillator, any frequency in the allowable frequency range can be realized, see [3]. Such a linear combination is slightly different in spirit from Eq. 2, since the coefficients are independent of time, and can be seen as a soft mixture of experts. We have the following result.

**Theorem 1** *Let $a < b$ be two arbitrary real numbers and let $f$ be a continuous function on $[0, T]$. Then for any error level $\epsilon > 0$, there exist $n$ and a function $g_n$ of the form*

$$g_n(t) = \sum_{k=1}^{n} \alpha_k e^{2\pi i \lambda_k t}, \qquad a \le \lambda_1, \ldots, \lambda_n \le b$$

*such that the uniform distance $\|f - g_n\|_\infty$ is less than $\epsilon$.*

In fact, it is not even necessary to vary the frequencies $\lambda$ over a continuous band $[a, b]$. We have the following.

**Theorem 2** *Let $\{\lambda_k\}$ be an infinite sequence with a finite accumulation point, and let $f$ be a continuous function on $[0, T]$. Then for any error level $\epsilon > 0$, there exist $n$ and a function $g_n(t) = \sum_{k=1}^{n} \alpha_k e^{2\pi i \lambda_k t}$ such that $\|f - g_n\|_\infty < \epsilon$.*

Thus, we may even fix the oscillator frequencies as e.g. $\lambda_k = 1/k$ without losing universal approximation capabilities. Similar statements can be made about mean-square approximation or, more generally, approximation in $p$-norm $L^p(\mu)$, where $1 \leq p < \infty$ and $\mu$ is a finite measure on $[0, T]$:

**Theorem 3** *For all $p$ and $f$ in $L^p(\mu)$ and for all $\epsilon > 0$, we can always find $n$ and $g_n$ as above such that $\|f - g_n\|_{L^p(\mu)} < \epsilon$.*

The proof of these results is surprisingly simple. Following the proofs in [4], if one of the above statements was not true, there would exist a nonzero, signed finite measure $\sigma$ with support in $[0, T]$ such that $\int_{[0,T]} e^{2\pi i \lambda t} \, d\sigma(t) = 0$ for all "allowed" frequencies $\lambda$. Now the function $z \mapsto \int_{[0,T]} e^{2\pi i z t} \, d\sigma(t)$ is clearly analytic on the whole complex plane. Hence, by a well-known result from complex variables, if it vanishes along an infinite sequence with a finite accumulation point, it is identically zero. But then in particular the Fourier transform of $\sigma$ vanishes, which in turn implies that $\sigma$ is identically zero by the uniqueness theorem on Fourier transforms, contradicting the initial assumption.

Notice that the above results do not imply that $f$ can exactly be represented as e.g. $f(t) = \int_a^b e^{2\pi i \lambda t} \, d\nu(\lambda)$ for some signed finite measure $\nu$—such functions are not only band-limited, but also extremely smooth (they have an analytic extension to the whole complex plane).

Hence, one might even conjecture that the above approximations are rather poor in the sense that unrealistically many terms are needed for the approximation. However, this is not true—one can easily show that the *rates of approximation cannot be worse that those for approximation with polynomials*. Let us briefly sketch the argument, because it also shows how bounded-frequency oscillators could be constructed.

Following an idea essentially due to Stinchcombe & White [5], let, more generally, $g$ be an analytic function in a neighborhood of the real line for which no derivative vanishes at the origin (above, we had $g(t) = e^{2\pi i t}$). Pick a nonnegative integer $n$ and a polynomial $p$ of degree not greater than $n - 1$ arbitrarily. Let us show that for any $\epsilon > 0$, we can always find a $g_n$ of the form $g_n(t) = \sum_{k=1}^{n} \alpha_k g(\lambda_k t)$ with $\lambda_k$ arbitrarily small such that $\|p - g_n\|_\infty < \epsilon$. To do so, note that we can write

$$g(\lambda t) = \sum_{l=0}^{n-1} \beta_l (\lambda t)^l + r_n(\lambda t), \qquad p(t) = \sum_{l=0}^{n-1} \delta_l t^l,$$

where $r_n(\lambda t)$ is of the order of $\lambda^n$, as $\lambda \to 0$, uniformly for $t$ in $[0, T]$. Hence,

$$\sum_{k=1}^{n} \alpha_k g(\lambda_k t) = \sum_{k=1}^{n} \alpha_k \left( \sum_{l=0}^{n-1} \beta_l (\lambda t)^l + r_n(\lambda t) \right)$$

$$= \sum_{l=0}^{n-1} \left( \sum_{k=1}^{n} \alpha_k \lambda_k^l \right) \beta_l t^l + \sum_{k=1}^{n} \alpha_k r_n(\lambda_k t).$$

Now fix $n$ distinct numbers $\ell_1, \ldots, \ell_n$, let $\lambda_k = \lambda_k(\rho) = \rho \ell_k$, and choose the $\alpha_k = \alpha_k(\rho)$ such that $\sum_{k=1}^{n} \alpha_k(\rho) \lambda_k(\rho)^l = \delta_l/\beta_l$ for $l = 0, \ldots, n-1$. (This is possible because, by assumption, all $\beta_l$ are non-zero.) It is readily seen that $\alpha_k(\rho)$ is of the order of $\rho^{1-n}$ as $\rho \to 0$ (in fact, the $j$-th row of the inverse of the coefficient matrix of the linear system is given by the coefficients of the polynomial $\prod_{k \neq j} (\lambda - \lambda_k)/(\lambda_j - \lambda_k)$). Hence, as $\rho \to 0$, the remainder term $\sum_{k=1}^{n} \alpha_k(\rho) r_n(\lambda_k(\rho)t)$ is of the order of $\rho$, and thus $\sum_{k=1}^{n} \alpha_k(\rho) g(\lambda_k(\rho)t) \to \sum_{l=0}^{n-1} \delta_l t^l = p(t)$ uniformly on $[0, T]$.

Note that using the above method, the coefficients in the approximation grow quite rapidly when the approximation error tends to 0. In some sense, this was to be

expected from the observation that the classes of small-band-limited functions are rather "small". There is a fundamental tradeoff between the size of the frequencies, and the size of the mixing coefficients. How exactly the coefficients scale with the width of the allowed frequency band is currently being investigated.

## 4 CONCLUSION

The modular oscillator approach leads to trajectory architectures which are more structured than fully interconnected networks, with a general feed-forward flow of information and sparse recurrent connections to achieve dynamical effects. The sparsity of units and connections are attractive features for hardware design; and so is also the modular organization and the fact that learning is much more circumscribed than in fully interconnected systems. We have shown in different ways that such architectures have universal approximation properties. In these architectures, however, some form of learning remains essential, for instance to fine tune each one of the modules. This, in itself, is a much easier task than the one a fully interconnected and random network would have been faced with. It can be solved by gradient or random descent or other methods. Yet, fundamental open problems remain in the overall organization of learning across modules, and in the origin of the decomposition. In particular, can the modular architecture be the outcome of a simple internal organizational process rather than an external imposition and how should learning be coordinated in time and across modules (other than the obvious: modules in the first level learn first, modules in the second level second, ...)? How successful is a global gradient descent strategy applied across modules? How can the same modular architecture be used for different trajectories, with short switching times between trajectories and proper phases along each trajectory?

**Acknowledgments**

The work of PB is in part supported by grants from the ONR and the AFOSR.

**References**

[1] Pierre Baldi. A modular hierarchical approach to learning. In *Proceedings of the 2nd International Conference on Fuzzy Logic and Neural Networks*, volume II, pages 985–988, IIzuka, Japan, 1992.

[2] Pierre F. Baldi. Gradient descent learning algorithm overview: a general dynamic systems perspective. *IEEE Transactions on Neural Networks*, 6(1):182–195, January 1995.

[3] Pierre F. Baldi and Amir F. Atiya. How delays affect neural dynamics and learning. *IEEE Transactions on Neural Networks*, 5(4):612–621, July 1994.

[4] Kurt Hornik. Some new results on neural network approximation. *Neural Networks*, 6:1069–1072, 1993.

[5] Maxwell B. Stinchcombe and Halbert White. Approximating and learning unknown mappings using multilayer feedforward networks with bounded weights. In *International Joint Conference on Neural Networks*, volume III, pages 7–16, Washington, 1990. Lawrence Earlbaum, Hillsdale.

## Footnotes

*Also with the Jet Propulsion Laboratory, California Institute of Technology.

[1] All the results to be derived can be extended immediately to the case of higher-dimensional trajectories.

[2] In a feed-forward network, where the transfer functions of the units are continuous, the output is a continuous function of the parameters and therefore there are no bifurcations.

[3]In what follows, we use the complex form for notational convenience.
